# Learning Saccadic Eye Movements Using Multiscale Spatial Filters

**Rajesh P.N. Rao and Dana H. Ballard**
Department of Computer Science
University of Rochester
Rochester, NY 14627
{rao,dana}@cs.rochester.edu

## Abstract

We describe a framework for learning saccadic eye movements using a photometric representation of target points in natural scenes. The representation takes the form of a high-dimensional vector comprised of the responses of spatial filters at different orientations and scales. We first demonstrate the use of this response vector in the task of locating previously foveated points in a scene and subsequently use this property in a multisaccade strategy to derive an adaptive motor map for delivering accurate saccades.

## 1 Introduction

There has been recent interest in the use of space-variant sensors in active vision systems for tasks such as visual search and object tracking [14]. Such sensors realize the simultaneous need for wide field-of-view and good visual acuity. One popular class of space-variant sensors is formed by *log-polar sensors* which have a small area near the optical axis of greatly increased resolution (the fovea) coupled with a peripheral region that witnesses a gradual logarithmic falloff in resolution as one moves radially outward. These sensors are inspired by similar structures found in the primate retina where one finds both a peripheral region of gradually decreasing acuity and a circularly symmetric *area centralis* characterized by a greater density of receptors and a disproportionate representation in the optic nerve [3]. The peripheral region, though of low visual acuity, is more sensitive to light intensity and movement.

The existence of a region optimized for discrimination and recognition surrounded by a region geared towards detection thus allows the image of an object of interest detected in the outer region to be placed on the more analytic center for closer scrutiny. Such a strategy however necessitates the existence of (a) methods to determine which location in the periphery to foveate next, and (b) fast gaze-shifting mechanisms to achieve this

foveation. In the case of humans, the "where-to-look-next" issue is addressed by both bottom-up strategies such as motion or salience clues from the periphery as well as top-down strategies such as search for a particular form or color. Gaze-shifting is accomplished via very rapid eye movements called *saccades*. Due to their high velocities, guidance through visual feedback is not possible and hence, saccadic movement is preprogrammed or ballistic: a pattern of muscle activation is calculated in advance that will direct the fovea almost exactly to the desired position [3].

In this paper, we describe an iconic representation of scene points that facilitates top-down foveal targeting. The representation takes the form of a high-dimensional vector comprised of the responses of different order Gaussian derivative filters, which are known to form the *principal components of natural images* [5], at variety of orientations and scales. Such a representation has been recently shown to be useful for visual tasks ranging from texture segmentation [7] to object indexing using a sparse distributed memory [11]. We describe how this photometric representation of scene points can be used in locating previously foveated points when a log-polar sensor is being used. This property is then used in a simple learning strategy that makes use of multiple corrective saccades to adaptively form a retinotopic motor map similar in spirit to the one known to exist in the deep layers of the primate superior colliculus [13]. Our approach differs from previous strategies for learning motor maps (for instance, [12]) in that we use the visual modality to actively supply the necessary reinforcement signal required during the motor learning step (Section 3.2).

## 2  The Multiscale Spatial Filter Representation

In the active vision framework, vision is seen as subserving a larger context of the encompassing behaviors that the agent is engaged in. For these behaviors, it is often possible to use temporary, iconic descriptions of the scene which are only relatively insensitive to variations in the view. Iconic scene descriptions can be obtained, for instance, by employing a bank of linear spatial filters at a variety of orientations and scales. In our approach, we use derivative of Gaussian filters since these are known to form the dominant eigenvectors of natural images [5] and can thus be expected to yield reliable results when used as basis functions for indexing[1].

The exact number of Gaussian derivative basis functions used is motivated by the need to make the representations invariant to rotations in the image plane (see [11] for more details). This invariance can be achieved by exploiting the property of *steerability* [4] which allows filter responses at arbitrary orientations to be synthesized from a finite set of basis filters. In particular, our implementation uses a minimal basis set of two first-order directional derivatives at $0°$ and $90°$, three second-order derivatives at $0°$, $60°$ and $120°$, and four third-order derivatives oriented at $0°$, $45°$, $90°$, and $135°$.

The response of an image patch $I$ centered at $(x_0, y_0)$ to a particular basis filter $G_i^{\theta_j}$ can be obtained by convolving the image patch with the filter :

$$r_{i,j}(x_0, y_0) = (G_i^{\theta_j} * I)(x_0, y_0) = \iint G_i^{\theta_j}(x_0 - x, y_0 - y) I(x, y) dx\, dy \qquad (1)$$

The iconic representation for the local image patch centered at $(x_0, y_0)$ is formed by combining into a single high-dimensional vector the responses from the nine basis filters, each (in our current implementation) at five different scales:

$$\vec{r}(x_0, y_0) = (r_{i,j,s}), \quad i = 1, 2, 3; j = 1, \ldots, i+1; s = s_{min}, \ldots, s_{max} \tag{2}$$

where $i$ denotes the order of the filter, $j$ denotes the number of filters per order, and $s$ denotes the number of different scales.

The use of multiple scales increases the perspicuity of the representation and allows interpolation strategies for scale invariance (see [9] for more details). The entire representation can be computed using only nine convolutions done at frame-rate within a pipeline image processor with nine constant size $8 \times 8$ kernels on a five-level octave-separated low-pass-filtered pyramid of the input image.

The 45-dimensional vector representation described above shares some of the favorable matching properties that accrue to high-dimensional vectors (cf. [6]). In particular, the distribution of distances between points in the 45-dimensional space of these vectors approximates a normal distribution; most of the points in the space lie at approximately the mean distance and are thus relatively uncorrelated to a given point [11]. As a result, the multiscale filter bank tends to generate almost unique location-indexed signatures of image regions which can tolerate considerable noise before they are confused with other image regions.

## 2.1 Localization

Denote the response vector from an image point as $\vec{r}_i$ and that from a previously foveated model point as $\vec{r}_m$. Then one metric for describing the similarity between the two points is simply the square of the Euclidean distance (or the sum-of-squared-differences) between their response vectors $d_{im} = \|\vec{r}_i - \vec{r}_m\|^2$. The algorithm for locating model points in a new scene can then be described as follows:

1. For the response vector representing a model point $m$, create a *distance image* $I_m$ defined by

   $$I_m(x, y) = min\left[I_{max} - \beta d_{im}, 0\right] \tag{3}$$

   where $\beta$ is a suitably chosen constant (this makes the best match the brightest point in $I_m$).

2. Find the best match point $(x_{b_m}, y_{b_m})$ in the image using the relation

   $$(x_{b_m}, y_{b_m}) = argmax\{I_m(x, y)\} \tag{4}$$

Figure 1 shows the use of the localization algorithm for targeting the optical axis of a uniform-resolution sensor in an example scene.

## 2.2 Extension to Space-Variant Sensing

The localization algorithm as presented above will obviously fail for sensors exhibiting nonuniform resolution characteristics. However, the multiscale structure of the response vectors can be effectively exploited to obtain a modified localization algorithm. Since decreasing radial resolution results in an effective reduction in scale (in addition to some

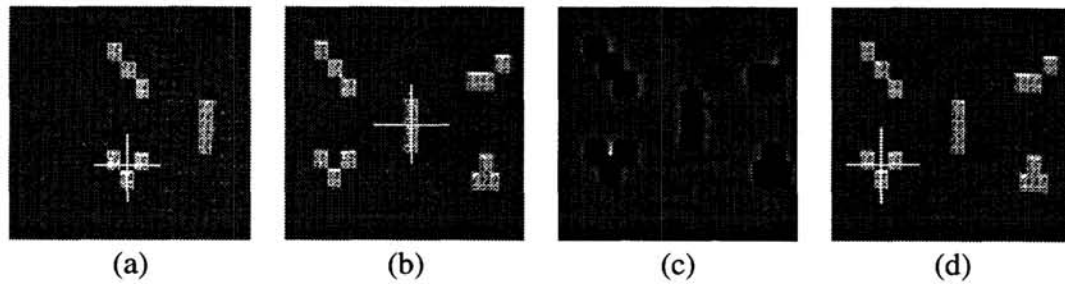

<div align="center">(a)              (b)              (c)              (d)</div>

Figure 1: Using response vectors to saccade to previously foveated positions. (a) Initial gaze point. (b) New gaze point; (c) To get back to the original point, the "distance image" is computed: the brightest spot represents the point whose response vector is closest to that of the original gaze point; (d) Location of best match is marked and an oculomotor command at that location can be executed to foveate that point.

other minor distortions) of previously foveated regions as they move towards the periphery, the filter responses previously occuring at larger scales now occur at smaller scales. Responses usually vary smoothly between scales; it is thus possible to establish a correspondence between the two response vectors of the same point on an object imaged at different scales by using a simple *interpolate-and-compare* scale matching strategy. That is, in addition to comparing an image response vector and a model response vector directly as outlined in the previous section, scale interpolated versions of the image vector are also compared with the original model response vector. In the simplest case, interpolation amounts to shifting image response vectors by one scale and thus, responses from a new image are compared with original model responses at second, third, ... scales, then with model responses at third, fourth, ... scales, and so on upto some threshold scale. This is illustrated in Figure 2 for two discrete movements of a simulated log-polar sensor.

## 3   The Multisaccade Learning Strategy

Since the high speed of saccades precludes visual guidance, advance knowledge of the precise motor command to be sent to the extraocular muscles for fixation of a desired retinal location is required. Results from neurophysiological and psychophysical studies suggest that in humans, this knowledge is acquired via learning: infants show a gradual increase in saccadic accuracy during their first year [1, 2] and adults can adapt to changes (caused for example by weakening of eye-muscles) in the interrelation between visual input and the saccades needed for centering. An adaptive mechanism for automatically learning the transfer function from retinal image space into motor space is also desirable in the context of active vision systems since an autonomous calibration of the saccadic system would (a) avoid the need for manual calibration, which can sometimes be complicated, and (b) provide resilience amidst changing circumstances caused by, for instance, changes in the camera lens mechanisms or degradation of the motor apparatus.

### 3.1   Motor Maps

In primates, the *superior colliculus* (SC), a multilayered neuron complex located in the upper regions of the brain stem, is known to play a crucial role in the saccade generation [13]. The upper layers of the SC contain a *retinotopic sensory map* with inputs from

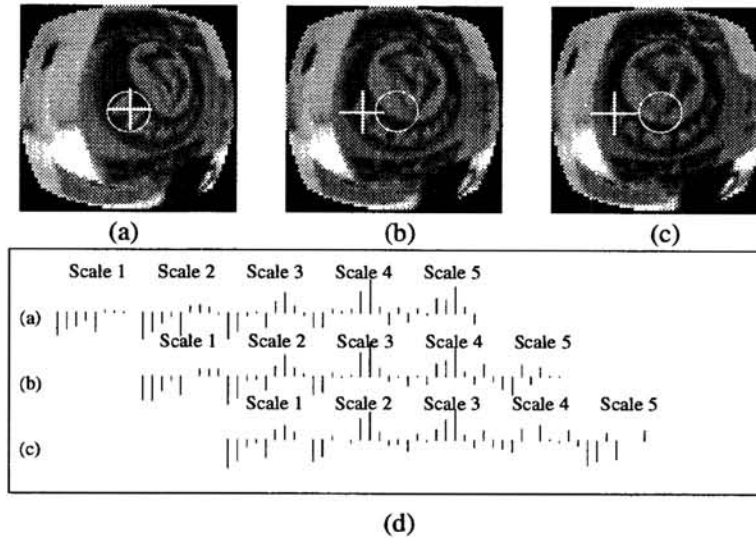

Figure 2: Using response vectors with a log-polar sensor. (a) through (c) represent a sequence of images (in Cartesian coordinates) obtained by movement of a simulated log-polar sensor from an original point (marked by '+') in the foveal region (indicated by a circle) towards the right. (d) depicts the process of interpolating (in this case, shifting) and matching response vectors of the same point as it moves towards the periphery of the sensor (Positive responses are represented by proportional upward bars and negative ones by proportional downward bars with the nine smallest scale responses at the beginning and the nine largest ones at the end).

the retina while the deeper layers contain a *motor map* approximately aligned with the sensory map. The motor map can be visualized as a topologically-organized network of neurons which reacts to a local activation caused by an input signal with a vectorial output quantity that can be transcoded into a saccadic motor command.

The alignment of the sensory and motor maps suggests the following convenient strategy for foveation: an excitation in the sensory layer (signaling a foveal target) is transferred to the underlying neurons in the motor layer which deliver the required saccade. In our framework, the excitation in the sensory layer before a goal-directed saccade corresponds to the brightest spot (most likely match) in the distance image (Figure 1 (c) for example). The formation of sensory map can be achieved using Kohonen's well-known stochastic learning algorithm by using a Gaussian input density function as described in [12]. Our primary interest lies not in the formation of the sensory map but in the development of a learning algorithm that assigns appropriate motor vectors to each location in the corresponding retinotopically-organized motor map. In particular, our algorithm employs a *visual reinforcement signal* obtained using iconic scene representations to determine the error vector during the learning step.

## 3.2 Learning the Motor Map

Our multisaccade learning strategy is inspired by the following observations in [2]: During the first few weeks after birth, infants appear to fixate randomly. At about 3 months of age, infants are able to fixate stimuli albeit with a number of corrective saccades of relatively large dispersion. There is however a gradual decrease in both the dispersion

and the number of saccades required for foveation in subsequent months (Figure 3 (a) depicts a sample set of fixations). After the first year, saccades are generally accurate, requiring at most one corrective saccade[2].

The learning method begins by assigning random values to the motor vectors at each location. The response vector for the current fixation point is first stored and a random saccade is executed to a different point. The goal then is to refixate the original point with the help of the localization algorithm and a limited number of multiple corrective saccades. The algorithm keeps track of the motor vector with minimum error during each run and updates the motor vectors for the neighborhood around the original unit whenever an improvement is observed. The current run ends when either the original point was successfully foveated or the limit $MAX$ for the maximum number of allowable corrective saccades was exceeded. A more detailed outline of the algorithm is as follows:

1. Initialize the motor map by assigning random values (within an appropriate range) to the saccadic motor vectors at each location. Align the optical axis of the sensor so that a suitable salient point falls on the fovea. Initialize the run number to $t := 0$.

2. Store in memory the filter response vector of the point $p$ currently in the center of the foveal region. Let $t := t + 1$.

3. Execute a random saccade to move the fovea to a different location in the scene.

4. Use the localization algorithm described in Section 2.2 and the stored response vector to find the location $l$ of the previously foveated point in the current retinal image. Execute a saccade using the motor vector $\vec{s}_l$ stored in this location in the motor map.

5. If the currently foveated region contains the original point $p$, return to 2 ($\vec{s}_l$ is accurate); otherwise,

   (a) Initialize the number of corrective saccades $N := 0$ and let $\vec{s} := \vec{s}_l$.

   (b) Determine the new location $l'$ of $p$ in the new image as in (4) and let $\vec{e}_{min}$ be the error vector, i.e. the vector from the foveal center to $l'$, computed from the output of the localization algorithm.

   (c) Execute a saccade using the motor vector $\vec{s}_{l'}$ stored at $l'$ and let $\vec{e}$ be the error vector (computed from the output of the localization algorithm) from the foveal center to the new location $l''$ of point $p$ found as in 4. Let $N := N + 1$ and let $\vec{s} := \vec{s} + \vec{s}_{l'}$ .

   (d) If $\|\vec{e}\| < \|\vec{e}_{min}\|$, then let $\vec{e}_{min} := \vec{e}$ and update the motor vectors for the units $k$ given by the neighborhood function $N(l, t)$ according to the well-known Kohonen rule:

   $$\vec{s}_k := \vec{s}_k + \gamma(t)(\vec{s} - \vec{s}_k) \tag{5}$$

   where $\gamma(t)$ is an appropriate gain function ($0 < \gamma(t) < 1$).

   (e) If the currently foveated region contains the original point $p$, return to 2; otherwise, if $N < MAX$, then determine the new location $l'$ of $p$ in the new image as in (4) and go to 5(c) (i.e. execute the next saccade); otherwise, return to 2.

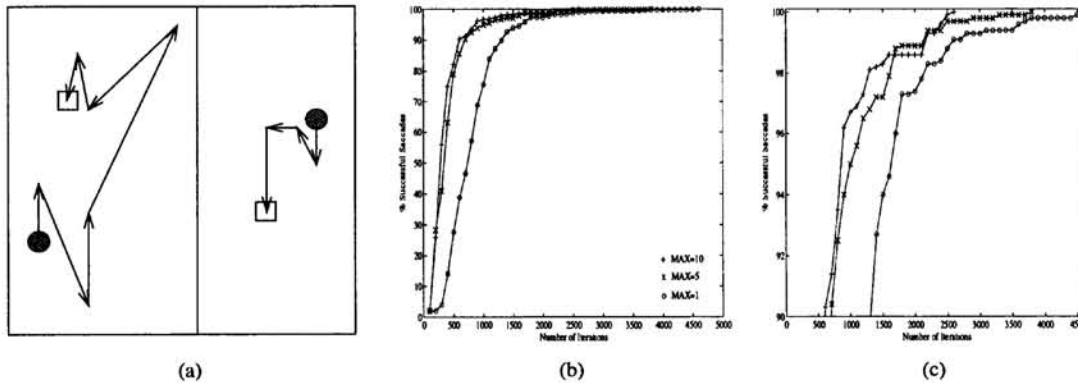

(a)  (b)  (c)

Figure 3: (a) Successive saccades executed by a 3-month old (left) and a 5-month old (right) infant when presented with a single illuminated stimulus (Adapted from [2]). (b) Graph showing % of saccades that end directly in the fovea plotted against the number of iterations of the learning algorithm for different values of $MAX$. (c) An enlarged portion of the same graph showing points when convergence was achieved.

The algorithm continues typically until convergence or the completion of a maximum number of runs. The gain term $\gamma(t)$ and the neighborhood $N(l, t)$ for any location $l$ are gradually decreased with increasing number of iterations $t$.

## 4    Results and Discussion

The simulation results for learning a motor map comprising of 961 units are shown in Figures 3 (b) and (c) which depict the variation in saccadic accuracy with the number of iterations of the algorithm for values of $MAX$ (maximum number of corrective saccades) of 1, 5 and 10. From the graphs, it can be seen that starting with an initially random assignment of vectors, the algorithm eventually assigns accurate saccadic vectors to all units. Fewer iterations seem to be required if more corrective saccades are allowed but then each iteration itself takes more time.

The localization algorithm described in Section 2.1 has been implemented on a *Datacube MaxVideo 200* pipeline image processing system and takes 1-2 seconds for location of points. Current work includes the integration of the multisaccade learning algorithm described above with the Datacube implementation and further evaluation of the learning algorithm. One possible drawback of the proposed algorithm is that for large retinal spaces, learning saccadic motor vectors for every retinal location can be time-consuming and in some cases, even infeasible [1]. In order to address this problem, we have recently proposed a variation of the current learning algorithm which uses a *sparse motor map* in conjunction with *distributed coding* of the saccadic motor vectors. This organization bears some striking similarities to Kanerva's sparse distributed memory model [6] and is in concurrence with recent neurophysiological evidence [8] supporting a distributed population encoding of saccadic movements in the superior colliculus. We refer the interested reader to [10] for more details.

**Acknowledgments**

We thank the NIPS*94 referees for their helpful comments. This work was supported by NSF research grant no. CDA-8822724, NIH/PHS research grant no. 1 R24 RRO6853, and a grant from the Human Science Frontiers Program.

## Footnotes

[1]In addition, these filters are endorsed by recent physiological studies [15] which show that derivative-of-Gaussians provide the best fit to primate cortical receptive field profiles among the different functions suggested in the literature.

[2]Large saccades in adults are usually *hypometric* i.e. they undershoot, necessitating a slightly slower corrective saccade. There is currently no universally accepted explanation for the need for such a two-step strategy.

# References

[1] Richard N. Aslin. Perception of visual direction in human infants. In C. Granlund, editor, *Visual Perception and Cognition in Infancy*, pages 91–118. Hillsdale, NJ: Lawrence Erlbaum Associates, 1993.

[2] Gordon W. Bronson. *The Scanning Patterns of Human Infants: Implications for Visual Learning*. Norwood, NJ: Ablex, 1982.

[3] Roger H.S. Carpenter. *Movements of the Eyes*. London: Pion, 1988.

[4] William T. Freeman and Edward H. Adelson. The design and use of steerable filters. *IEEE Transactions on Pattern Analysis and Machine Intelligence*, 13(9):891–906, September 1991.

[5] Peter J.B. Hancock, Roland J. Baddeley, and Leslie S. Smith. The principal components of natural images. *Network*, 3:61–70, 1992.

[6] Pentti Kanerva. *Sparse Distributed Memory*. Bradford Books, Cambridge, MA, 1988.

[7] Jitendra Malik and Pietro Perona. A computational model of texture segmentation. In *IEEE Conference on Computer Vision and Pattern Recognition*, pages 326–332, June 1989.

[8] James T. McIlwain. Distributed spatial coding in the superior colliculus: A review. *Visual Neuroscience*, 6:3–13, 1991.

[9] Rajesh P.N. Rao and Dana H. Ballard. An active vision architecture based on iconic representations. Technical Report 548, Department of Computer Science, University of Rochester, 1995.

[10] Rajesh P.N. Rao and Dana H. Ballard. A computational model for visual learning of saccadic eye movements. Technical Report 558, Department of Computer Science, University of Rochester, January 1995.

[11] Rajesh P.N. Rao and Dana H. Ballard. Object indexing using an iconic sparse distributed memory. Technical Report 559, Department of Computer Science, University of Rochester, January 1995.

[12] Helge Ritter, Thomas Martinetz, and Klaus Schulten. *Neural Computation and Self-Organizing Maps: An Introduction*. Reading, MA: Addison-Wesley, 1992.

[13] David L. Sparks and Rosi Hartwich-Young. The deep layers of the superior colliculus. In R.H. Wurtz and M.E. Goldberg, editors, *The Neurobiology of Saccadic Eye Movements*, pages 213–255. Amsterdam: Elsevier, 1989.

[14] Massimo Tistarelli and Giulio Sandini. Dynamic aspects in active vision. *Computer Vision, Graphics, and Image Processing: Image Understanding*, 56(1):108–129, 1992.

[15] R.A. Young. The Gaussian derivative theory of spatial vision: Analysis of cortical cell receptive field line-weighting profiles. *General Motors Research Publication GMR-4920*, 1985.